# Multiple Instance Filtering

**Kamil Wnuk**          **Stefano Soatto**

University of California, Los Angeles

{kwnuk,soatto}@cs.ucla.edu

## Abstract

We propose a robust filtering approach based on semi-supervised and multiple instance learning (MIL). We assume that the posterior density would be unimodal if not for the effect of outliers that we do not wish to explicitly model. Therefore, we seek for a point estimate at the outset, rather than a generic approximation of the entire posterior. Our approach can be thought of as a combination of standard finite-dimensional filtering (Extended Kalman Filter, or Unscented Filter) with multiple instance learning, whereby the initial condition comes with a putative set of inlier measurements. We show how both the state (regression) and the inlier set (classification) can be estimated iteratively and causally by processing only the current measurement. We illustrate our approach on visual tracking problems whereby the object of interest (target) moves and evolves as a result of occlusions and deformations, and partial knowledge of the target is given in the form of a bounding box (training set).

## 1 Introduction

Algorithms for filtering and prediction have a venerable history studded by quantum leaps by Wiener, Kolmogorov, Mortensen, Zakai, Duncan among others. Many attempts to expand finite-dimensional optimal filtering beyond the linear-Gaussian case failed,[1] which explains in part the resurgence of general-purpose approximation methods for the filtering equation, such as weak-approximations (particle filters [6, 16]) as well as parametric ones (*e.g.*, sum-of-Gaussians or interactive multiple models [5]). Unfortunately, in many applications of interest, from visual tracking to robotic navigation, the posterior is not unimodal. This has motivated practitioners to resort to general-purpose approximations of the entire posterior, mostly using particle filtering. However, in many applications one has reason to believe that the posterior would be unimodal if not for the effect of *outlier* measurements, and therefore the interest is in a *point estimate*, for instance the mode, mean or median, rather than in the entire posterior. So, we tackle the problem of filtering, where the data is partitioned into two unknown subsets (inliers and outliers). Our goal is to devise finite-dimensional filtering schemes that will approximate the dominant mode of the posterior distribution, without explicitly modeling the outliers. There is a significant body of related work, summarized below.

### 1.1 Prior related work

Our goal is naturally framed in the classical robust statistical inference setting, whereby classification (inlier/outlier) is solved along with regression (filtering). We assume that an initial condition is available, both for the regressor (state) as well as the inlier distribution.

The latter can be thought of as training data in a semi-supervised setting. Robust filtering has been approached from many perspectives: Using a robust norm (typically $H^\infty$ or $\ell^1$) for the prediction residual yields worst-case disturbance rejection [14, 9]; rejection sampling schemes in the spirit of the M-estimator [11] "robustify" classical filters and their extensions. These approaches work with few outliers, say $10-20\%$, but fail in vision applications where one typically has 90% or more. Our approach relates to recent work in detection-based tracking [3, 10] that use semi-supervised learning [4, 18, 13], as well as multiple-instance learning [2] and latent-SVM models [8, 20].

In [3] an ensemble of pixel-level weak classifiers is combined on-line via boosting; this is efficient but suffers from drift; [10] improves stability by using a static model trained on the first frame as a prior for labeling new training samples used to update an online classifier. MILTrack [4] addressed the problem of selecting training data for model update so as to maintain maximum discriminative power. This is related to our approach, except that we have an explicit dynamical model, rather than a scanning window for detection. Also, our discrimination criterion operates on a collection of parts/regions rather than a single template. This allows more robustness to deformations and occlusions. We adopt an incremental SVM with a fast approximation of a nonlinear kernel [21] rather than online boosting. Our part based representation and explicit dynamics allow us to better handle scale and shape changes without the need for a multi-scale image search [4, 13]. PROST [18] proposed a cascade of optical flow, online random forest, and template matching. The P-N tracker [13] combined a median flow tracker with an online random forest. New training samples were collected when detections violated structural constraints based on estimated object position. In an effort to control drift, new training data was not incorporated into the model until the tracked object returned to a previously confirmed appearance with high confidence. This meant that if object appearance never returned to the "key frames," the online model would never be updated. In the aforementioned works objects are represented as a bounding box. Several recent approaches have also used segmentation to improve the reliability of tracking: [17] did not leverage temporal information beyond adjacent frames, [22] required several annotated input frames with detailed segmentations, and [7] relied on trackable points on both sides of the object boundary. In all methods above there was no explicit temporal modeling beyond adjacent frames; therefore the schemes had poor predictive capabilities. Other approaches have used explicit temporal models together with sparsity constraints to model appearance changes [15].

We propose a semi-supervised approach to filtering, with an explicit temporal model, that assumes imperfect labeling, whereby portions of the image inside the bounding box are "true positives" and others are outliers. This enables us to handle appearance changes, for instance due to partial occlusions or changes of vantage point.

## 1.2   Formalization

We denote with $x(t) \in \mathbb{R}^n$ the state of the model at time $t \in \mathbb{Z}^+$. It describes a discrete-time trajectory in a finite-dimensional (vector) space. This can be thought of as a realization of a stochastic process that evolves via some kind of ordinary difference equation $x(t+1) = f(x(t)) + \nu(t)$, where $\nu(t) \overset{IID}{\sim} p_\nu$ is a temporally independent and identically distributed process. We will assume that, possibly after whitening, the components of $\nu(t)$ are independent.

We denote the set of measurements at time $t$ with $y(t) = \{y_i(t)\}_{i=1}^{m(t)}$, $y_i(t) \in \mathbb{R}^k$. We assume each can be represented by some fixed dimensionality descriptor, $\phi : \mathbb{R}^k \to \mathbb{R}^l; (y) \to \phi(y)$. In classical filtering, the measurements are a known function of the state, $y(t) = h(x(t)) + n(t)$, up to the measurement noise, $n(t)$, that is a realization of a stochastic process that is often assumed to be temporally independent and identically distributed, and also independent of $\nu(t)$. In our case, however, the components of the measurement process $y_1(t), \ldots, y_{m(t)}(t)$ are divided into two groups: those that behave like standard measurements in a filtering process, and those that do not.

This distinction is made by an indicator variable $\chi(t) \in \{-1, 1\}^{m(t)}$ of the same dimensionality as the number of measurements, whose values are unknown, and can change over time.

For brevity of notation we denote the two sets of indexes as $\chi(t)^+ = \{i \mid \chi_i(t) = 1\}$ and $\chi(t)^- = \{i \mid \chi_i(t) = -1\}$. For the first set we have that $\{y_i(t)\}_{i \in \chi(t)^+} = h(x(t), t) + n(t)$, just like in classical filtering, except that the measurement model $h(\cdot, t)$ is time-varying in a way that includes singular perturbations, since the number of measurements changes over time, so the function $h : \mathbb{R}^n \times \mathbb{R} \to \mathbb{R}^{m(t)}$; $(x, t) \mapsto h(x, t)$ changes dimension over time. For the second group, unlike particle filtering, we do not care to model their states, and instead just discount them as *outliers*. The measurements are thus samples from a stochastic process that includes two independent sources of uncertainty: the measurement noise, $n(t)$, and the selection process $\chi(t)$.

Our goal is that of determining a point-estimate of the state $x(t)$ given measurements up to time $t$. This will be some statistic (the mean, median, mode, etc.) of the conditional density $p(x(t)|\{y(k)\}_{k=1}^t)$, where the process $\chi(t)$ has to be marginalized.

In order to design a filter, we first consider the full forward model of how the various samples of the inlier measurements are generated. To this end, we assume that the inlier set is separable from the outlier set by a hyper-plane in some feature space, represented by the normal vector $w(t) \in \mathbb{R}^l$. So, given the assignment of inliers and outliers $\chi(t)$, we have that the new maximal-margin boundary can be obtained from $w(t-1)$ by several iterations of a stochastic subgradient descent procedure [19], which for brevity we denote as $w(t) = \text{stochSubgradIters}(w(t-1), y(t), \chi(t))$ and describe in Sec. 2 and Sec. 2.2. Conversely, if we are given the hyperplane $w(t)$, and state $x(t)$, the measurements can be classified via $\chi(t) = \text{argmin}_\chi E(y(t), w(t), x(t), \chi)$. The energy function, $E(y(t), w(t), x(t), \chi)$ depends on how one chooses to model the object and what *side information* is applied to constrain the selection of training data. In the implementation details we give examples of how appearance continuity can be used as a constraint in this step. Further, motion similarity and occlusion boundaries could also be used.

Finally, the forward (data-formation) model for a sample (realization) of the measurement process is given as follows: At time $t = 0$, we will assume that we have available an initial distribution $p(x_0)$ together with an initial assignment of inliers and outliers $\chi_0$, so $x(0) \sim p(x_0)$; $\chi(0) = \chi_0$. Given $\chi(0)$, we bootstrap our classifier by minimizing a standard support vector machine cost function: $w(1) = \text{argmin}_w(\frac{\lambda}{2}||w||^2 + \frac{1}{m(0)} \sum_{i=1}^{m(0)} \max(0, 1 - \chi_i(0))\langle w, \phi(y_i(0))\rangle)$, where $\lambda \in \mathbb{R}$ is the tradeoff between the importance of margin size versus loss. At all subsequent times $t$, each realization evolves according to:

$$\begin{cases} x(t+1) = f(x(t)) + v(t), \\ w(t+1) = \text{stochSubgradIters}(w(t), y(t), \chi(t)), \\ \chi(t) = \text{argmin}_\chi E(y(t), w(t), x(t), \chi), \\ \{y_i(t)\}_{i \in \chi(t)^+} = h(x(t), t) + n(t). \end{cases} \quad (1)$$

where the first two equations can be thought of as the "model equations" and the last two as the "measurement equations." The presence of $\chi_0$ makes this a semi-supervised learning problem, where $\chi_0$ is the "training set" for the process $\chi(t)$. Note that it is possible for the model above to proceed in open-loop, when no inliers are present.

The model (1) can easily be extended to the case when the measurement equation is in implicit form, $h(x(t), \{y_i(t)\}_{i \in \chi(t)^+}, t) = n(t)$, since all that matters is the innovation process $e(t) \doteq h(\{y_i(t)\}_{i \in \chi(t)^+}, \hat{x}(t), t)$. Additional extensions can be entertained where the dynamics $f$ depends on the classifier $w$, so that $x(t+1) = f(x(t), w(t)) + v(t)$, and similarly for the measurement equation $h(x(t), w(t), t)$, although we will not consider them here.

## 1.3 Application example: Visual tracking with shape and appearance changes

Objects of interest (e.g. humans, cars) move in ways that result in a deformation of their projection onto the image plane, even when the object is rigid. Further changes of appearance occur due to motion relative to the light source and partial occlusions. Because of the ambiguities in shape and appearance, one can fix one factor and model the other. For instance, one can fix a bounding box (shape) and model change of appearance inside,

including outliers (due to occlusion) and inliers (newly visible portions of the object). Alternatively, one can enforce constancy of the reflectance function, but then shape changes as well as illumination must be modeled explicitly, which is complex [12].

Our approach tracks the motion of a bounding box, enclosing the data inliers. Call $c(t) \in \mathbb{R}^2$ the center of this bounding box, $v_c(t) \in \mathbb{R}^2$ the velocity of the center, $d(t) \in \mathbb{R}^2$ the length of the sides of the bounding box, and $v_d(t) \in \mathbb{R}^2$ its rate of change. Thus, we have $x(t) = [c(t), v_c(t), d(t), v_d(t)]^T$. As before $\chi(t)$ indicates a binary labeling of the measurement components, where $\chi(t)^+$ is the set of samples that correspond to the object of interest. We have tested different versions of our framework where the components are *superpixels* as well as *trajectories of feature points*. For reasons of space limitation, below we describe the case of superpixels, and report results for trajectories as supplementary material.

Consider a time-varying image $I(t) : D \subset \mathbb{R}^2 \to \mathbb{R}^+; (u, v) \mapsto I(u, v, t)$: superpixels $\{S_i\}$ are just a partition of the domain $D = \cup_{i=1}^r S_i$ with $S_i \cap S_j = \delta_{ij}$; $\chi(t)$ becomes a binary labeling of the superpixels, with $\chi(t)^+$ collecting the indices of elements on the object of interest, and $\chi(t)^-$ on the background.

The measurement equation is obtained as the centroid and diameter of the restriction of the bounding box to the domain of the inlier super-pixels: If $y(t) = I(t) \in \mathbb{R}^{N \times M}$ is an image, then $h_1(\{I(u, v, t)\}_{(u,v) \in S_i}) \in \mathbb{R}^2$ is the centroid of the superpixels $\{S_i\}_{i \in \chi(t)^+}$ computed from $I(t)$, and $h_2(\{I(u, v, t)\}_{(u,v) \in S_i}) \in \mathbb{R}^2$ is the diameter of the same region. This is in the form (1), with $h$ constant (the time dependency is only through $y(t)$ and $\chi(t)$). The resulting model is:

$$\begin{cases} x(t+1) = Fx(t) + \nu(t) \\ w(t+1) = \text{stochSubgradIters}(w(t), y(t), \chi(t)) \\ \chi(t) = \text{argmin}_\chi E(y(t), w(t), x(t), \chi) \\ h(y_i(t)_{i \in \chi(t)^+}) = Cx(t) + n(t) \end{cases} \quad (2)$$

where $F \in \mathbb{R}^{8 \times 8}$ is block-diagonal with each $4 \times 4$ block given by $\begin{bmatrix} I & I \\ 0 & I \end{bmatrix}$, $C \in \mathbb{R}^{4 \times 8}$, $C = \begin{bmatrix} I & 0 & 0 & 0 \\ 0 & 0 & I & 0 \end{bmatrix}$, and $I$ is the $2 \times 2$ identity matrix. Similarly, $\nu(t) \overset{\text{IID}}{\sim} \mathcal{N}(0, Q)$, $Q \in \mathbb{R}^{8 \times 8}$ and $n(t) \overset{\text{IID}}{\sim} \mathcal{N}(0, R)$, $R \in \mathbb{R}^{4 \times 4}$.

## 2   Algorithm development

We focus our discussion in this section on the development of the discriminative appearance model at the heart of the inlier/outlier classification, $w(t)$. For simplicity, pretend for now that each frame contains $m$ observations. We assume an object is identified with a subset of the observations (inliers); at time $t$, we have $\{y_i(t)\}_{i \in \chi(t)^+}$. Also pretend that observations from all frames, $Y = \{y(t)\}_{t=1}^{N_f}$, were available simultaneously; $N_f$ is the number of frames in the video sequence. If all frames were labeled, ($\chi(t)$ known $\forall$ $t$), a maximum margin classifier $\hat{w}$ could be obtained by minimizing the objective (3) over all samples in all frames:

$$\hat{w} = \underset{w}{\text{argmin}} \left( \frac{\lambda}{2} ||w||^2 + \frac{1}{mN_f} \sum_{t=1}^{N_f} \sum_{i=1}^m \ell(w, \phi(y_i(t)), \chi_i(t)) \right). \quad (3)$$

where $\lambda \in \mathbb{R}$, and $\ell(w, \phi(y_i(t)), \chi_i(t))$ is a loss that ensures data fit. We use the hinge loss $\ell(w, \phi(y_i(t)), \chi_i(t)) = \max(0, 1 - \chi_i(t)\langle w, \phi(y_i(t))\rangle)$ in which slack is implicit, so we can use an efficient sequential optimization in the primal form.

In reality an exact label assignment at every frame is not available, so we must infer the latent labeling $\chi$ simultaneously while learning the hyperplane $w$. Continuing our hypothetical batch processing scenario, pretend we have estimates of some state of the object throughout time, $\hat{X} = \{\hat{x}(t)\}_{t=1}^{N_f}$. This allows us to identify a reduced subset of candidate inliers

(in MIL terminology a *positive bag*), within which we assume all inliers are contained. The specification of a positive bag helps reduce the search space, since we can assume all samples outside of a positive bag are negative. This changes the SVM formulation to a mixed integer program similar to the mi-SVM [2], except that [2] assumed a positive/negative bag partition was given, whereas we use the estimated state and add a term to the decision boundary cost function to express the dependence between the labeling, $\chi(t)$, and state estimate, $\hat{x}$, at each time:

$$\hat{w}, \hat{\chi} = \underset{w,\chi}{\operatorname{argmin}} \left( \frac{\lambda}{2}||w||^2 + \frac{1}{mN_f} \sum_{t=1}^{N_f} \left( \sum_{i=1}^{m} \max\left(0, 1 - \chi_i(t)\langle w, \phi(y_i(t))\rangle\right) + E\left(y(t), \chi, \hat{x}(t)\right) \right) \right).$$
(4)

Here $E(y(t), \chi(t), \hat{x}(t))$ represents a general mechanism to enforce constraints on label assignment on a per-frame basis within a temporal sequence.[2] A standard optimization procedure alternates between updating the decision boundary $w$, subject to an estimated labeling $\hat{\chi}$, followed by relabeling the original data to satisfy the positive bag constraints generated from the state estimates, $\hat{x}$, while keeping $w$ fixed:

$$\begin{cases} \hat{w} = \operatorname{argmin}_w \left( \frac{\lambda}{2}||w||^2 + \frac{1}{mN_f} \sum_{t=1}^{N_f} \sum_{i=1}^{m} \max(0, 1 - \hat{\chi}_i(t)\langle w, \phi(y_i(t))\rangle) \right), \\ \hat{\chi} = \operatorname{argmin}_\chi \frac{1}{mN_f} \sum_{t=1}^{N_f} \left( \sum_{i=1}^{m} \max(0, 1 - \chi_i(t)\langle \hat{w}, \phi(y_i(t))\rangle) + E(y(t), \chi(t), \hat{x}(t)) \right). \end{cases}$$
(5)

In practice, annotation is available only in the first frame, and the data must be processed causally and sequentially. Recently, [19] proposed an efficient incremental scheme, PEGA-SOS, to solve the hinge loss objective in the primal form. This enables straightforward incremental training of $w$ as new data becomes available. The algorithm operates on a training set consisting of tuples of labeled descriptors: $T = \{(\phi(y_i), \chi_i)\}_{i=1}^m$. In a nutshell, at each PEGASOS iteration we select a subset of training samples from the current training set $A_j \subseteq T$, and update $w$ according to $w_{j+1} = w_j - \eta_j \nabla_j$. The subgradient of the hinge loss is given by $\nabla_j = \lambda w_j - \frac{1}{|A_j|} \sum_{i \in A_j} \chi_i \phi(y_i)$. To finalize the update and accelerate convergence $w_{j+1}$ is projected onto the set $\{w : ||w|| \leq \frac{1}{\sqrt{\lambda}}\}$, which [19] show is the space containing the optimal solution.

The second objective of Eq. (5) seeks a solution to the binary integer program of inlier selection given $\hat{w}$ and $\hat{x}$. Instead of tackling this NP-hard problem, we re-interpret it as a constraint enforcement step based on additional cues within a search area specified by our the current state estimate. One example constraint for a superpixel based object representation is to re-interpret the given objective as a graph cut problem, with pairwise terms enforcing appearance consistency. See supplementary material for details, as well as for experiments with other choices of constraints for tracks, rather than superpixels.

## 2.1 Initialization

At $t = 0$ we are given initial observations $y(0)$ and a bounding box indicating the object of interest $\{c(0) \pm d(0)\}$. We initialize $\chi(0)$ with positive indices corresponding to superpixels that have a majority of their area $|y_i(0)|$ within the bounding box:

$$\chi_i(0) = \begin{cases} 1 \text{ if } \frac{|\{c(0)\pm d(0)\} \cap y_i(0)|}{|y_i(0)|} > \epsilon_y, \\ -1 \text{ otherwise.} \end{cases}$$
(6)

The area threshold is $\epsilon_y = 0.7$ throughout all experiments. This represents a bootstrap training set, $T_1$ from which we learn an initial classifier $w(1)$ for distinguishing object appearance. Each element of the training set is a triplet $(\phi(y_i(t)), \chi_i(t), \tau_i = t)$, where the last element is the time at which the feature is added to the training set. We start by selecting all positive samples and a set number of negatives, $n_f$, sampled randomly from $\chi(0)^-$, giving $T_1 = \{(\phi(y_i(0)), \chi_i(0), 0)\}_{\forall i \in \chi(0)^+} \cup \{(\phi(y_j(0)), \chi_j(0), 0) \mid j \in \chi(0)_{\text{rand}}^- \subseteq \chi(0)^-, |\chi(0)_{\text{rand}}^-| = n_f\}$.

## 2.2 Prediction Step

At time $t$, given the current estimate of the object state and classification $\chi(t)$, we add all positive samples and difficult negative samples lying outside of the estimated bounding box to the new training set $T_{t+1|t}$. We then propagate the object state with the model of motion dynamics and finally update the decision boundary with the newly updated training set.

$$
\begin{cases}
\hat{x}(t+1|t) & = F\hat{x}(t|t) \\
P(t+1|t) & = FP(t|t)F^T + Q \\
T_{t+1} & = T_{t+1,\text{old}} \cup T_{t+1,\text{new}} \\
T_{t+1,\text{old}} & = \{(\phi(y_i),\chi_i,\tau_i) \mid \chi_i\langle\phi(y_i),w(t)\rangle < 1,\ t-\tau_i \leq \tau_{\max}\} \\
T_{t+1,\text{new}} & = \{(\phi(y_i(t)),\chi_i(t),t) \mid \chi_i(t)=1\} \cup \\
& \quad \{(\phi(y_i(t)),-1,t) \mid \frac{|D/\{\hat{c}(t|t)\pm\hat{d}(t|t)\}\ \cap\ y_i(t)|}{|y_i(t)|} \geq 1-\epsilon_y,\ \langle\phi(y_i(t)),w(t)\rangle > -1\} \\
w(t+1) & \leftarrow \text{for } j = n_T,...,N \quad (\text{update starting with } w_{n_T}=w(t)) \\
& \qquad \text{choose } A_j \subseteq T_{t+1} \\
& \qquad n_j = \frac{1}{\lambda j} \\
& \qquad w_{j+1} = (1-\eta_j\lambda)w_j + \frac{\eta_j}{|A_j|}\sum_{i\in A_j}\chi_i(t)\phi(y_i(t)) \\
& \qquad w_{j+1} = \min\{1,\frac{1/\sqrt{\lambda}}{||w_{j+1}||}\}w_{j+1} \\
& \quad \text{end}
\end{cases}
\tag{7}
$$

It is typically not necessary to update $w$ at every step, so training data can be collected over several frames during which $w(t+1) = w(t)$ and the update above can be invoked either at some regular interval, on demand, or upon some form of model validation as in [13]. The parameter $\tau_{\max}$ determines memory of the classifier update procedure for difficult examples. If $\tau_{\max} = 0$, no memory is used and training data for model update consists only of observations from the current image. Such a memory of recent training samples is analogous to the training cache used in [8] for training the latentSVM model. During each classifier update we perform $N - n_T$ iterations of the stochastic subgradient descent algorithm, starting from the current best estimate of the separating hyperplane $w_{n_T} = w(t)$. The overall number of iterations $N$ is set as $N = 20/\lambda$, where $\lambda$ is a function of the bootstrap training set size, $\lambda = 1/(10|T_1|)$. The number in the denominator is used as a parameter to set the relative importance of the margin size and the loss, but we fix it at 10 for our experiments. The number of iterations at a new time is then decided by $n_T = \max(1-|T_t|/N, 0.75)$ in order to limit how much the hyperplane can change in a single update. These parameters can also be viewed as tuning the learning rates and forgetting factors of the classifier.

## 2.3 Update Step

The innovation is in implicit form with $h(y_i(t+1)_{i\in\chi(t+1)^+}) \in \mathbb{R}^4$ giving a tight bounding box around the selected foreground regions in the same form as they appear in the state. In the update equations $r$ specifies the size of the search region around the predicted state within which we consider observations as candidates for foreground; $\xi$ specifies the indices of candidate observations (positive bag).

$$
\begin{cases}
r & = \lambda_r(\begin{bmatrix} I & 0 \end{bmatrix}\text{diag}(CP(t+1|t)C^T) + \begin{bmatrix} 0 & I \end{bmatrix}\text{diag}(CP(t+1|t)C^T), \\
\xi & = \{i \mid \frac{|\{c(t+1|t)\pm(d(t+1|t)+r)\}\ \cap\ y_i(t+1)|}{|y_i(t+1)|} > \mathcal{E}_y\}, \\
\chi(t+1) & = \text{argmin}_{\chi\in\{-1,1\}^m}\ E(w(t+1),\{y_i(t+1)\}_{i\in\xi},\hat{x}(t+1|t),\chi) \\
e(t+1) & = h(y_i(t+1)_{i\in\chi(t+1)^+}) - C\hat{x}(t+1|t) \\
L & = P_{t+1|t}C^T(CP_{t+1|t}C^T + R)^{-1} \\
\hat{x}(t+1|t+1) & = \hat{x}(t+1|t) + Le(t+1) \\
P(t+1|t+1) & = (I-LC)P(t+1|t)(I-LC)^T + LRL^T.
\end{cases}
\tag{8}
$$

Above $\lambda_r \in \mathbb{R}$ is a factor (we fix it at 3) for scaling the region size based on filter covariance.

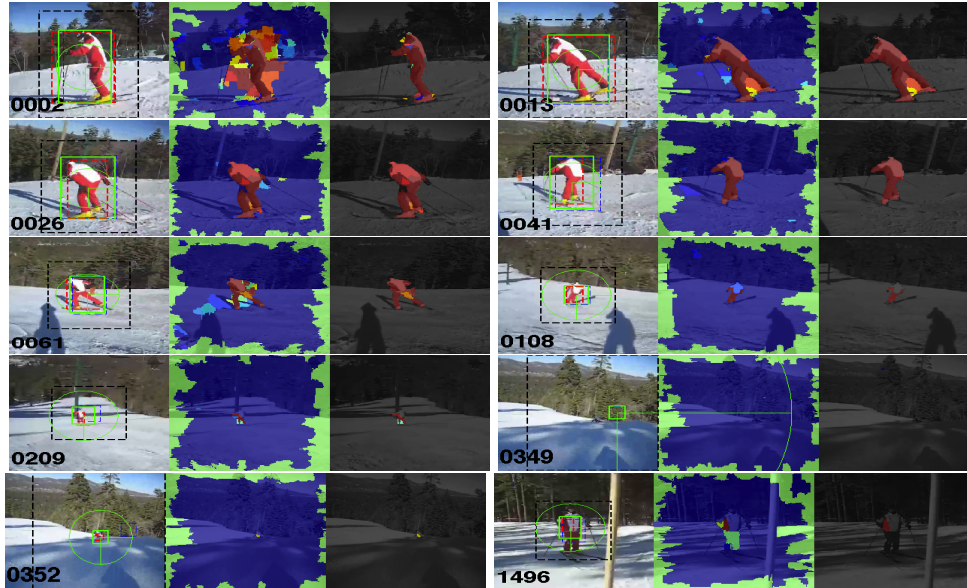

Figure 1: *Ski sequence: Left panel shows frame number, search area (black rectangle), filter prediction (blue), observation (red), and updated filter estimate (green). The center panels overlay the SVM scores for each region (solid blue = −1, solid red = 1). Right panels show the regions selected as inliers. This challenging sequence includes viewpoint and scale changes, deformation, changing background. The algorithm performs well and successfully recovers from missed detection (from frame 349 to 352 shown above).*

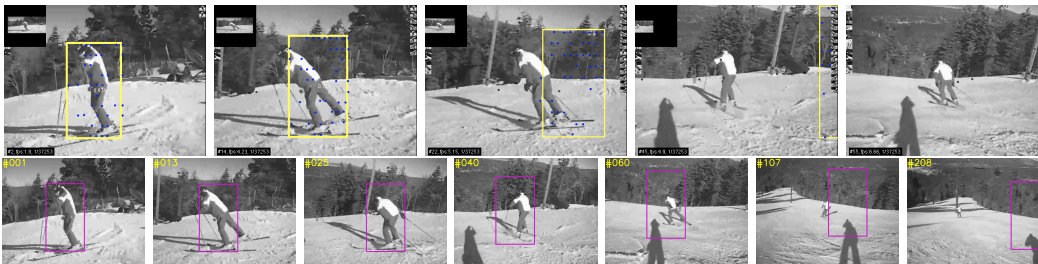

Figure 2: *P-N tracker [13] (above) and MILTrack [4] (below) initialized with the same bounding box as our approach. Original implementations by the respective authors were used for this comparison. The P-N tracker fails because of the absence of stable low-level tracks on the target and quickly locks onto a patch of trees in the background. MILTrack survives longer but does not adapt scale quickly enough, eventually drifting to become a detector of the tree line.*

## 3   Experiments

To compare with [18, 4, 13], we first evaluate our discriminative model without maintaining any training data history $\tau_{\max} = 0$ and updating $w$ every 6 frames, with training data collected between incremental updates. Even with $\tau_{\max} = 0$ we can track highly deforming objects (a skier) with significant scale changes through most of the 1496 frames (Fig. 1). We also recover from errors due to the implicit memory in the decision boundary from incremental updating. For comparison, [4, 13] quickly drift and fail to recover (Fig. 2).

For a quantitative comparison we test our full algorithm against the state of the art on the PROST dataset [18] consisting of 4 videos with fast motion, occlusions, scale changes, translucency, and small background motions. In all experiments $\tau_{\max} = 25$, and all other parameters were fixed as described earlier and in supplementary material. Two evaluation metrics are reported: the mean center location error in pixels [4], and percentage of correctly tracked frames as computed by the bounding box overlap criteria $\frac{area(ROI_D \cap ROI_{GT})}{area(ROI_D \cup ROI_{GT})} > 0.5$,

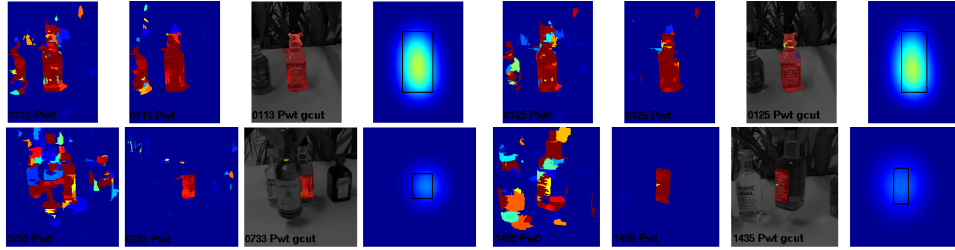

Figure 3: *Convergence of the classifier: Samples from frames 113, 125, 733, and 1435 of the "liquor" sequence. The leftmost image shows the probabilities returned by the initial classifier trained using only the first frame, the second image shows the foreground probabilities returned from the current classifier, the third image shows the foreground selection made by the graph-cut step, and the final image shows the smoothed score used to select bounding box location.*

where $ROI_D$ is the detected region and $ROI_{GT}$ is the ground truth region. The ground truth for the PROST dataset is reported using a constant sized bounding box. Table 1 compares to [18, 4, 1, 13].

In the liquor sequence our method correctly shrinks the bounding box to the label, since the rest of the bottle is not discriminative. Unfortunately, this is penalized in the Pascal score since the area ratio drops below 0.5 of the initial bounding box despite perfect tracking. This causes the score to drop to 18.9. If we modify the criterion to count as valid a detection where $> 99\%$ of the detection area lies within the annotated ground truth region, the score becomes 75.6%. If we allow for $> 90\%$ of the detected area to lie within the ground truth box, the final pascal result for the liquor sequence becomes 79.1%. See Figure 3. The same phenomenon occurs in the box sequence, where our approach adapts to tracking the label at the bottom of the box. Note, this additional detection criteria has no effect on any other scores. Additional results, including failure modes as well as successful tracking where other approaches fail, are reported in the supplementary material, both for the case of superpixels and tracks.

|  | Overall | board | | box | | lemming | | liquor | |
|---|---|---|---|---|---|---|---|---|---|
|  | pascal | pascal | distance | pascal | distance | pascal | distance | pascal | distance |
| ours | 74.7 | **92.1** | **13.7** | 42.9* | 63.7 | **88.1** | 19.4 | 75.6* | 42.5* |
| P-N [13] | 37.15 | 12.9 | 139.5 | 36.9 | 99.3* | 34.3 | 26.4* | 64.5 | 17.4* |
| PROST [18] | 80.4 | 75.0 | 39.0 | **90.6** | **13.0** | 70.5 | 25.1 | **85.4** | **21.5** |
| MILTrack [4] | 49.2 | 67.9 | 51.2 | 24.5 | 104.6 | 83.6 | **14.9** | 20.6 | 165.1 |
| FragTrack [1] | 66.0 | 67.9 | 90.1 | 61.4 | 57.4 | 54.9 | 82.8 | 79.9 | 30.7 |

Table 1: *Comparison with recent methods on the PROST dataset. Best scores for each sequence and metric are shown in bold. Our method and the P-N tracker [13] do not always detect the object. Ground truthed frames in which no location was reported by the method of [13] were not counted into the final distance score. The method of [13] missed 2 detections on the box sequence, 1 detection on the lemming sequence, and 80 on the liquor sequence. When our approach failed to detect the object, we used the predicted bounding box from the state of the filter as our reported result.*

## 4   Discussion

We have proposed an approach to robust filtering embedding a multiple instance learning SVM within a filtering framework, and iteratively performing regression (filtering) and classification (inlier selection) in hope of reaching an approximate estimate of the dominant mode of the posterior for the case where other modes are due to outlier processes in the measurements. We emphasize that our approach comes with no provable properties or guarantees, other than for the trivial case when the dynamics are linear, the inlier-outlier sets are linearly separable, the noises are Gaussian, zero-mean, IID white and independent with known covariance, and when the initial inlier set is known to include all inliers but is not necessarily pure. In this case, the method proposed converges to the conditional mean of the posterior $p(x(t)|\{y(k)\}_{k=1}^{t})$. However, we have provided empirical validation of our approach on challenging visual tracking problems, where it exceeds the state of the art, and illustrated some of its failure modes.

**Acknowledgment:** Research supported by AFOSR FA9550-09-1-0427, ONR N000141110863, and DARPA FA8650-11-1-7156.

## Footnotes

[1]Also due to the non-existence of invariant family of distributions for large classes of Fokker-Planck operators.

[2]It represents the *side information* necessary to avoid zero information gain in the semi-supervised inference procedure.

# References

[1] A. Adam, E. Rivlin, and I. Shimshoni. Robust fragments-based tracking using the integral histogram. In *Proc. CVPR*, 2006.

[2] S. Andrews, I. Tsochantaridis, and T. Hofmann. Support vector machines for multiple-instance learning. In *Proc. NIPS*, 2003.

[3] S. Avidan. Ensemble tracking. *PAMI*, 29:261–271, 2007.

[4] B. Babenko, M.-H. Yang, and S. Belongie. Visual tracking with online multiple instance learning. In *Proc. CVPR*, 2009.

[5] Y. Bar-Shalom and X.-R. Li. *Estimation and tracking: principles, techniques and software.* YBS Press, 1998.

[6] A. Doucet, N. de Freitas, and N. Gordon. *Sequential monte carlo methods in practice.* Springer Verlag, New York, 2001.

[7] J. Fan, X. Shen, and Y. Wu. Closed-loop adaptation for robust tracking. In *Proc. ECCV*, 2010.

[8] P. Felzenszwalb, D. Girshick, D. McAllester, and D. Ramanan. Object detection with discriminatively trained part based models. In *PAMI*, 2010.

[9] L. El Ghaoui and G. Calafiore. Robust filtering for discrete-time systems with structured uncertainty. In *IEEE Transactions on Automatic Control*, 2001.

[10] H. Grabner, C. Leistner, and H. Bischof. Semi-supervised on-line boosting for robust tracking. In *Proc. ECCV*, 2008.

[11] P.J. Huber. *Robust Statistics.* Wiley, New York, 1981.

[12] J. Jackson, A. J. Yezzi, and S. Soatto. Dynamic shape and appearance modeling via moving and deforming layers. *IJCV*, 79(1):71–84, August 2008.

[13] Z. Kalal, J. Matas, and K. Mikolajczyk. P-n learning: Bootstrapping binary classifiers by structural constraints. In *Proc. CVPR*, 2010.

[14] H. Li and M. Fu. A linear matrix inequality approach to robust h$\infty$ filtering. *IEEE Transactions on Signal Processing*, 45(9):2338–2350, September 1997.

[15] H. Lim, V. Morariu, O. Camps, and M. Sznaier. Dynamic appearance modeling for human tracking. In *Proc. CVPR*, 2006.

[16] J. Liu. *Monte carlo strategies in scientific computing.* SPringer Verlag, 2001.

[17] X. Ren and J. Malik. Tracking as repeated figure/ground segmentation. In *Proc. CVPR*, 2007.

[18] J. Santner, C. Leistner, A. Saffari, T. Pock, and H. Bischof. PROST Parallel Robust Online Simple Tracking. In *Proc. CVPR*, 2010.

[19] S. Shalev-Shwartz, Y. Singer, and N. Srebro. Pegasos: Primal estimated sub-gradient solver for svm. In *Proc. ICML*, 2007.

[20] I. Tsochantaridis, T. Joachims, T. Hofmann, and Y. Altun. Large margin methods for structured and interdependent output variables. *JMLR*, 6:1453–1484, September 2005.

[21] A. Vedaldi and A. Zisserman. Efficient additive kernels via explicit feature maps. In *Proc. CVPR*, 2010.

[22] Z. Yin and R. T. Collins. Shape constrained figure-ground segmentation and tracking. In *Proc. CVPR*, 2009.

